# Unsupervised Learning of Visual Sense Models for Polysemous Words

**Kate Saenko**
MIT CSAIL
Cambridge, MA
saenko@csail.mit.edu

**Trevor Darrell**
UC Berkeley EECS / ICSI
Berkeley, CA
trevor@eecs.berkeley.edu

## Abstract

Polysemy is a problem for methods that exploit image search engines to build object category models. Existing unsupervised approaches do not take word sense into consideration. We propose a new method that uses a dictionary to learn models of visual word sense from a large collection of unlabeled web data. The use of LDA to discover a latent sense space makes the model robust despite the very limited nature of dictionary definitions. The definitions are used to learn a distribution in the latent space that best represents a sense. The algorithm then uses the text surrounding image links to retrieve images with high probability of a particular dictionary sense. An object classifier is trained on the resulting sense-specific images. We evaluate our method on a dataset obtained by searching the web for polysemous words. Category classification experiments show that our dictionary-based approach outperforms baseline methods.

## 1 Introduction

We address the problem of unsupervised learning of object classifiers for visually polysemous words. Visual polysemy means that a word has several dictionary senses that are visually distinct. Web images are a rich and free resource compared to traditional human-labeled object datasets. Potential training data for arbitrary objects can be easily obtained from image search engines like Yahoo or Google. The drawback is that multiple word meanings often lead to mixed results, especially for polysemous words. For example, the query "mouse" returns multiple senses on the first page of results: "computer" mouse, "animal" mouse, and "Mickey Mouse" (see Figure 1.) The dataset thus obtained suffers from low precision of any particular visual sense.

Some existing approaches attempt to filter out unrelated images, but do not directly address polysemy. One approach involves bootstrapping object classifiers from labeled image data [9], others cluster the unlabeled images into coherent components [6],[2]. However, most rely on a labeled seed set of inlier-sense images to initialize bootstrapping or to select the right cluster. The unsupervised approach of [12] bootstraps an SVM from the top-ranked images returned by a search engine, with the assumption that they have higher precision for the category. However, for polysemous words, the top-ranked results are likely to include several senses.

We propose a fully unsupervised method that specifically takes word sense into account. The only input to our algorithm is a list of words (such as all English nouns, for example) and their dictionary entries. Our method is multimodal, using both web search images and the text surrounding them in the document in which they are embedded. The key idea is to learn a text model of the word sense, using an electronic dictionary such as Wordnet together with a large amount of unlabeled text. The model is then used to retrieve images of a specific sense from the mixed-sense search results. One application is an image search filter that automatically groups results by word sense for easier navigation for the user. However, our main focus in this paper is on using the re-ranked images

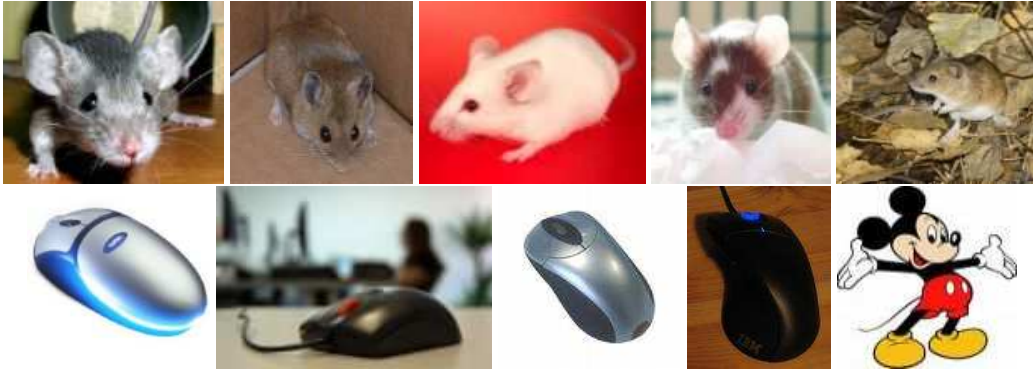

Figure 1: **Which sense of "mouse"?** Mixed-sense images returned from an image keyword search.

as training data for an object classifier. The resulting classifier can predict not only the English word that best describes an input image, but also the correct sense of that word.

A human operator can often refine the search by using more sense-specific queries, for example, "computer mouse" instead of "mouse". We explore a simple method that does this automatically by generating sense-specific search terms from entries in Wordnet (see Section 2.3). However, this method must rely on one- to three-word combinations and is therefore brittle. Many of the generated search terms are too unnatural to retrieve any results, e.g., "percoid bass". Some retrieve many unrelated images, such as the term "ticker" used as an alternative to "watch". We regard this method as a baseline to our main approach, which overcomes these issues by learning a model of each sense from a large amount of text obtained by searching the web. Web text is more natural and is a closer match to the text surronding web images than dictionary entries, which allows us to learn more robust models. Each dictionary sense is represented in the latent space of hidden "topics" learned empirically for the polysemous word.

To evaluate our algorithm, we collect a dataset by searching the Yahoo Search engine for five polysemous words: "bass", "face", "mouse", "speaker" and "watch". Each of these words has anywhere from three to thirteen noun senses. Experimental evaluation on this dataset includes both retrieval and classification of unseen images into specific visual senses.

## 2 Model

The inspiration for our method comes from the fact that text surrounding web images indexed by a polysemous keyword can be a rich source of information about the sense of that word. The main idea is to learn a probabilistic model of each sense, as defined by entries in a dictionary (in our case, Wordnet), from a large amount of unlabeled text. The use of a dictionary is key because it frees us from needing a labeled set of images to learn the visual sense model.

Since this paper is concerned with objects rather than actions, we restrict ourselves to entries for nouns. Like standard word sense disambiguation (WSD) methods, we make a one-sense-per-document assumption [14], and rely on words co-occurring with the image in the HTML document to indicate that sense. Our method consists of three steps: 1) discovering latent dimensions in text associated with a keyword, 2) learning probabilistic models of dictionary senses in that latent space, and 3) using the text-based sense models to construct sense-specific image classifiers. We will now describe each step in detail.

### 2.1 Latent Text Space

Unlike words in text commonly used in WSD, image links are not guaranteed to be surrounded by grammatical prose. This makes it difficult to extract structured features such as part-of-speech tags. We therefore take a bag-of-words approach, using all available words near the image link to evaluate the probability of the sense. The first idea is to use a large collection of such bags-of-words to learn coherent dimensions which align with different senses or uses of the word.

We could use one of several existing techniques to discover latent dimensions in documents consisting of bags-of-words. We choose to use Latent Dirichlet Allocation, or LDA, as introduced by Blei et. al.[4]. LDA discovers hidden topics, i.e. distributions over discrete observations (such as words), in the data. Each document is modeled as a mixture of topics $z \in \{1, ..., K\}$. A given collection of $M$ documents, each containing a bag of $N_d$ words, is assumed to be generated by the following process: First, we sample the parameters $\phi^j$ of a multinomial distribution over words from a Dirichlet prior with parameter $\beta$ for each topic $j = 1, ..., K$. Then, for each document $d$, we sample the parameters $\theta_d$ of a multinomial distribution over topics from a Dirichlet prior with parameter $\alpha$. Finally, for each word token $i$, we choose a topic $z_i$ from the multinomial $\theta_d$, and then choose a word $w_i$ from the multinomial $\phi^{z_i}$. The probability of generating a document is defined as

$$P(w_1, ..., w_{N_d}|\phi, \theta_d) = \prod_{i=1}^{N_d} \sum_{z=1}^{K} P(w_i|z, \phi)\ P(z|\theta_d) \tag{1}$$

Our initial approach was to learn hidden topics using LDA directly on the words surrounding the images. However, while the resulting topics were often aligned along sense boundaries, the approach suffered from over-fitting, due to the irregular quality and low quantity of the data. Often, the only clue to the image's sense is a short text fragment, such as "fishing with friends" for an image returned for the query "bass". To allieviate the overfitting problem, we instead create an additional dataset of text-only web pages returned from regular web search. We then learn an LDA model on this dataset and use the resulting distributions to train a model of the dictionary senses, described next.

## 2.2 Dictionary Sense Model

We use the limited text available in the Wordnet entries to relate dictionary sense to topics formed above. For example, sense 1 of "bass" contains the definition "the lowest part of the musical range." To these words we also add the synonyms (e.g., "pitch"), the hyponyms, if they exist, and the first-level hypernyms (e.g., "sound property"). We denote the bag-of-words extracted from such a dictionary entry for sense $s$ as $e_s = w_1, w_2, ..., w_{E_s}$, where $E_s$ is the number of words in the bag. The model is trained as follows: Given a query word with sense $s \in \{1, 2, ...S\}$ we define the likelihood of a particular sense given the topic $j$ as

$$P(s|z = j) \equiv \frac{1}{E_s} \sum_{i=1}^{E_s} P(w_i|z = j), \tag{2}$$

or the average likelihood of words in the definition. For a web image with an associated text document $d = w_1, w_2, ..., w_D$, the model computes the probability of a particular sense as

$$P(s|d) = \sum_{j=1}^{K} P(s|z = j)P(z = j|d). \tag{3}$$

The above requires the distribution of LDA topics in the text context, $P(z|d)$, which we compute by marginalizing across words and using Bayes' rule:

$$P(z = j|d) = \sum_{i=1}^{D} P(z = j|w_i) = \sum_{i=1}^{D} \frac{P(w_i|z = j)P(z = j)}{P(w_i)}, \tag{4}$$

and also normalizing for the length of the text context. Finally, we define the probability of a particular dictionary sense given the image to be equal to $P(s|d)$. Thus, our model is able to assign sense probabilities to images returned from the search engine, which in turn allows us to group the images according to sense.

## 2.3 Visual Sense Model

The last step of our algorithm uses the sense model learned in the first two steps to generate training data for an image-based classifier. The choice of classifier is not a crucial part of the algorithm. We choose to use a discriminative classifier, in particular, a support vector machine (SVM), because of its ability to generalize well in high-dimentional spaces without requiring a lot of training data.

Table 1: **Dataset Description:** sizes of the three datasets, and distribution of ground truth sense labels in the keyword dataset.

| category | size of datasets | | | distribution of labels in the keyword dataset | |
|---|---|---|---|---|---|
| | text-only | sense term | keyword | positive (good) | negative (partial, unrelated) |
| Bass | 984 | 357 | 678 | 146 | 532 |
| Face | 961 | 798 | 756 | 130 | 626 |
| Mouse | 987 | 726 | 768 | 198 | 570 |
| Speaker | 984 | 2270 | 660 | 235 | 425 |
| Watch | 936 | 2373 | 777 | 512 | 265 |

For each particular sense $s$, the model re-ranks the images according to the probability of that sense, and selects the $N$ highest-ranked examples as positive training data for the SVM. The negative training data is drawn from a "background" class, which in our case is the union of all other objects that we are asked to classify. We represent images as histograms of visual words, which are obtained by detecting local interest points and vector-quantizing their descriptors using a fixed visual vocabulary.

We compare our model with a simple baseline method that attempts to refine the search by automatically generating search terms from the dictionary entry. Experimentally, it was found that queries consisting of more than about three terms returned very few images. Consequently, the terms are generated by appending the polysemous word to its synonyms and first-level hypernyms. For example, sense 4 of "mouse" has synonym "computer mouse" and hypernym "electronic device", which produces the terms "computer mouse" and "mouse electronic device". An SVM classifier is then trained on the returned images.

## 3 Datasets

To train and evaluate the outlined algorithms, we use three datasets: image search results using the given keyword, image search results using sense-specific search terms, and text search results using the given keyword.

The first dataset was collected automatically by issuing queries to the Yahoo Image Search™ website and downloading the returned images and HTML web pages. The keywords used were: "bass", "face", "mouse", "speaker" and "watch". In the results, "bass" can refer to a fish or a musical term, as in "bass guitar"; "face" has a multitude of meanings, as in "human face", "animal face", "mountain face", etc.; "speaker" can refer to audio speakers or human speakers; "watch" can mean a timepiece, the act of watching, as in "hurricane watch", or the action, as in "watch out!" Samples that had dead page links and/or corrupted images were removed from the dataset.

The images were labeled by a human annotator with one sense per keyword. The annotator labeled the presense of the following senses: "bass" as in fish, "face" as in a human face, "mouse" as in computer mouse, "speaker" as in an audio output device, and "watch" as in a timepiece. The annotator saw only the images, and not the text or the dictionary definitions. The labels used were $0: unrelated$, $1: partial$, or $2: good$. Images where the object was too small or occluded were labeled $partial$. For evaluation, we used only $good$ labels as positive, and grouped $partial$ and $unrelated$ images into the negative class. The labels were only used in testing, and not in training.

The second image search dataset was collected in a similar manner but using the generated sense-specific search terms. The third, text-only dataset was collected via regular web search for the original keywords. Neither of these two datasets were labeled. Table 1 shows the size of the datasets and distribution of labels.

## 4 Features

When extracting words from web pages, all HTML tags are removed, and the remaining text is tokenized. A standard stop-word list of common English words, plus a few domain-specific words like "jpg", is applied, followed by a Porter stemmer [11]. Words that appear only once and the actual word used as the query are pruned. To extract text context words for an image, the image link is

located automatically in the corresponding HTML page. All word tokens in a 100-token window surrounding the location of the image link are extracted. The text vocabulary size used for the sense model ranges between 12K-20K words for different keywords.

To extract image features, all images are resized to 300 pixels in width and converted to grayscale. Two types of local feature points are detected in the image: edge features [6] and scale-invariant salient points. In our experiments, we found that using both types of points boosts classficiation performance relative to using just one type. To detect edge points, we first perform Canny edge detection, and then sample a fixed number of points along the edges from a distribution proportional to edge strength. The scales of the local regions around points are sampled uniformly from the range of 10-50 pixels. To detect scale-invariant salient points, we use the Harris-Laplace [10] detector with the lowest strength threshold set to 10. Altogether, 400 edge points and approximately the same number of Harris-Laplace points are detected per image. A 128-dimensional SIFT descriptor is used to describe the patch surrounding each interest point. After extracting a bag of interest point descriptors for each image, vector quantization is performed. A codebook of size 800 is constructed by k-means clustering a randomly chosen subset of the database (300 images per keyword), and all images are converted to histograms over the resulting visual words. To be precise, the "visual words" are the cluster centers (codewords) of the codebook. No spatial information is included in the image representation, but rather it is treated as a bag-of-words.

## 5 Experiments

### 5.1 Re-ranking Image Search Results

In the first set of experiments, we evaluate how well our text-based sense model can distinguish between images depicting the correct visual sense and all the other senses. We train a separate LDA model for each keyword on the text-only dataset, setting the number of topics $K$ to 8 in each case. Although this number is roughly equal to the average number of senses for the given keywords, we do not expect nor require each topic to align with one particular sense. In fact, multiple topics can represent the same sense. Rather, we treat $K$ as the dimensionality of the latent space that the model uses to represent senses. While our intuition is that it should be on the order of the number of senses, it can also be set automatically by cross-validation. In our initial experiments, different values of $K$ did not significantly alter the results.

To perform inference in LDA, a number of approximate inference algorithms can be applied. We use a Gibbs sampling approach of [7], implemented in the Matlab Topic Modeling Toolbox [13]. We used symmetric Dirichlet priors with scalar hyperparameters $\alpha = 50/K$ and $\beta = 0.01$, which have the effect of smoothing the empirical topic distribution, and 1000 iterations of Gibbs sampling.

The LDA model provides us with topic distributions $P(w|z)$ and $P(z)$. We complete training the model by computing $P(s|z)$ for each sense $s$ in Wordnet, as in Equation 2. We train a separate model for each keyword. We then compute $P(s|d)$ for all text contexts $d$ associated with images in the keyword dataset, using Equation 3, and rank the corresponding images according to the probability of each sense. Since we only have ground truth labels for a single sense per keyword (see Section 3), we evaluate the retrieval performance for that particular ground truth sense. Figure 2 shows the resulting ROCs for each keyword, computed by thresholding $P(s|d)$. For example, the first plot shows ROCs obtained by the eight models corresponding to each of the senses of the keyword "bass". The thick blue curve is the ROC obtained by the original Yahoo retrieval order. The other thick curves show the dictionary sense models that correspond to the ground truth sense (a fish). The results demonstrate that we are able to learn a useful sense model that retrieves far more positive-class images than the original search engine order. This is important in order for the first step of our method to be able to improve the precision of training data used in the second step. Note that, for some keywords, there are multiple dictionary definitions that are difficult to distinguish visually, for example, "human face" and "facial expression". In our evaluation, we did not make such fine-grained distinctions, but simply chose the sense that applied most generally.

In interactive applications, the human user can specify the intended sense of the word by providing an extra keyword, such as by saying or typing "bass fish". The correct dictionary sense can then be selected by evaluating the probability of the extra keyword under each sense model, and choosing the highest-scoring one.

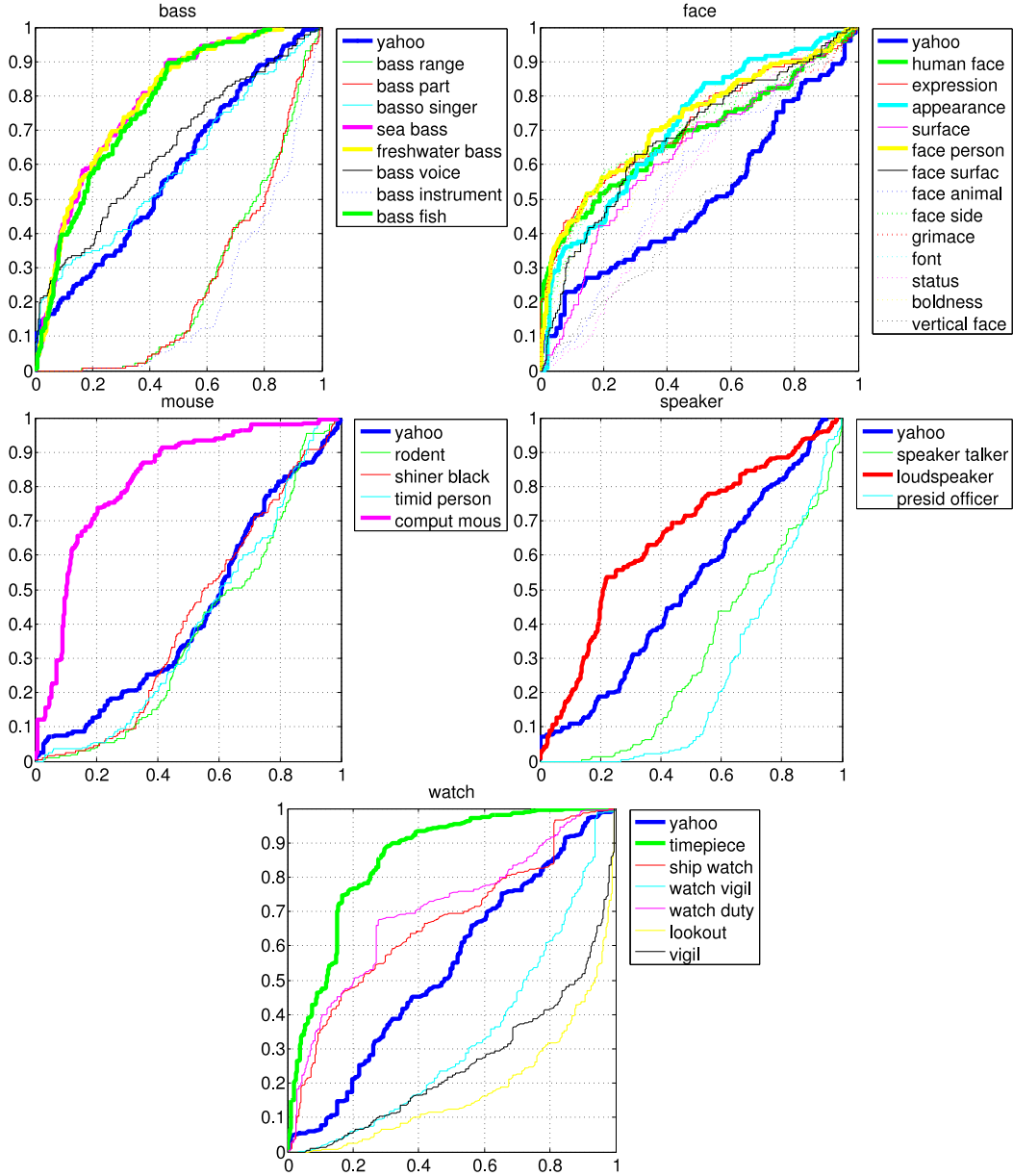

Figure 2: **Retrieval** of the ground truth sense from keyword search results. Thick blue lines are the ROCs for the original Yahoo search ranks. Other thick lines are the ROCs obtained by our dictionary model for the true senses, and thin lines are the ROCs obtained for the other senses.

## 5.2 Classifying Unseen Images

The goal of the second set of experiments is to evaluate the dictionary-based object classifier. We train a classifier for the object corresponding to the ground-truth sense of each polysemous keyword in our data. The clasifiers are binary, assigning a positive label to the correct sense and a negative label to incorrect senses and all other objects. The top N unlabeled images ranked by the sense model are selected as positive training images. The unlabeled pool used in our model consists of both the keyword and the sense-term datasets. N negative images are chosen at random from positive data for all other keywords. A binary SVM with an RBF kernel is trained on the image features, with the $C$ and $\gamma$ parameters chosen by four-fold cross-validation. The baseline search-terms algorithm that we compare against is trained on a random sample of N images from the sense-term dataset. Recall

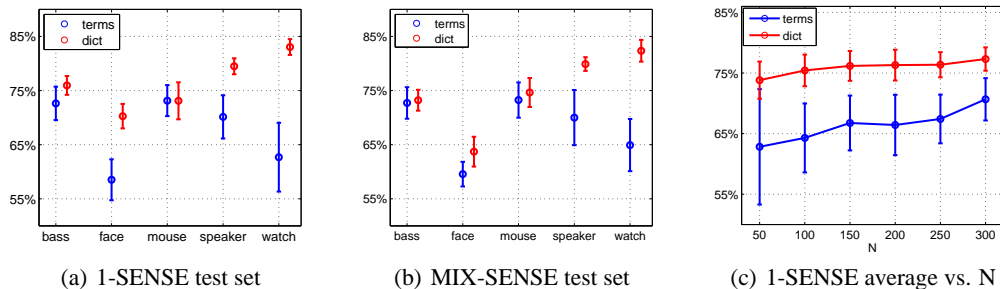

| (a) 1-SENSE test set | (b) MIX-SENSE test set | (c) 1-SENSE average vs. N |

Figure 3: **Classification accuracy** for the search-terms baseline (terms) and our dictionary model (dict).

that this dataset was collected by simply searching with word combinations extracted from the target sense definition. Training on the first N images returned by Yahoo did not qualitatively change the results.

We evaluate the method on two test cases. In the first case, the negative class consists of only the ground-truth senses of the other objects. We refer to this as the 1-SENSE test set. In the second case, the negative class also includes other senses of the given keyword. For example, we test detection of "computer mouse" among other keyword objects as well as "animal mouse", "Mickey Mouse" and other senses returned by the search, including unrelated images. We refer to this as the MIX-SENSE test set. Figure 3 compares the classification accuracy of our classifier to the baseline search-terms classifier. Average accuracy across ten trials with different random splits into train and test sets is shown for each object. Figure 3(a) shows results on 1-SENSE and 3(b) on MIX-SENSE, with N equal to 250. Figure 3(c) shows 1-SENSE results averaged over the categories, at different numbers of training images $N$. In both test cases, our dictionary method significantly improves on the baseline algorithm. As the per-object results show, we do much better for three of the five objects, and comparably for the other two. One explanation why we do not see a large improvement in the latter cases is that the automatically generated sense-specific search terms happened to return relatively high-precision images. However, in the other three cases, the term generation fails while our model is still able to capture the dictionary sense.

# 6   Related Work

A complete review of WSD work is beyond the scope of the present paper. Yarowsky [14] proposed an unsupervised WSD method, and suggested the use of dictionary definitions as an initial seed.

Several approaches to building object models using image search results have been proposed, although none have specifically addressed polysemous words. Fei-Fei et. al. [9] bootstrap object classifiers from existing labeled image data. Fergus et. al. [6] cluster in the image domain and use a small validation set to select a single positive component. Schroff et. al. [12] incorporate text features (such as whether the keyword appears in the URL) and use them re-rank the images before training the image model. However, the text ranker is *category-independent* and does not learn which words are predictive of a specific sense. Berg et. al. [2] discover topics using LDA in the *text* domain, and then use them to cluster the images. However, their method requires manual intervention by the user to sort the topics into positive and negative for each category. The combination of image and text features is used in some web retrieval methods (e.g. [5]), however, our work is focused not on instance-based image retrieval, but on *category-level* modeling.

A related problem is modeling images annotated with words, such as the caption "sky, airplane", which are assigned by a human labeler. Barnard et. al. [1] use visual features to help disambiguate word senses in such loosely labeled data. Models of annotated images assume that there is a correspondence between each image region and a word in the caption (e.g. Corr-LDA, [3]). Such models predict words, which serve as category labels, based on image content. In contrast, our model predicts a category label based on all of the words in the web image's text context. In general, a text context word does not necessarily have a corresponding visual region, and vice versa.

In work closely related to Corr-LDA, a People-LDA [8] model is used to guide topic formation in news photos and captions, using a specialized face recognizer. The caption data is less constrained than annotations, including non-category words, but still far more constrained than text contexts.

## 7   Conclusion

We introduced a model that uses a dictionary and text contexts of web images to disambiguate image senses. To the best of our knowledge, it is the first use of a dictionary in either web-based image retrieval or classifier learning. Our approach harnesses the large amount of unlabeled text available through keyword search on the web in conjunction with the dictionary entries to learn a generative model of sense. Our sense model is purely unsupervised, and is appropriate for web images. The use of LDA to discover a latent sense space makes the model robust despite the very limited nature of dictionary definitions. The definition text is used to learn a distribution over the empirical text topics that best represents the sense. As a final step, a discriminative classifier is trained on the re-ranked mixed-sense images that can predict the correct sense for novel images.

We evaluated our model on a large dataset of over 10,000 images consisting of search results for five polysemous words. Experiments included retrieval of the ground truth sense and classification of unseen images. On the retrieval task, our dictionary model improved on the baseline search engine precision by re-ranking the images according to sense probability. On the classification task, our method outperformed a baseline method that attempts to refine the search by generating sense-specific search terms from Wordnet entries. Classification also improved when the test objects included the other senses of the keyword, making distinctions such as "loudspeaker" vs. "invited speaker". Of course, we would not expect the dictionary senses to always produce accurate visual models, as many senses do not refer to objects (e.g. "bass voice"). Future work will include annotating the data with more senses to further explore the "visualness" of some of them.

## References

[1] K. Barnard, K. Yanai, M. Johnson, and P. Gabbur. Cross modal disambiguation. In Toward Category-Level Object Recognition, J. Ponce, M. Hebert, C. Schmidt, eds., Springer-Verlag LNCS Vol. 4170, 2006.

[2] T. Berg and D. Forsyth. Animals on the web. In Proc. CVPR, 2006.

[3] D. Blei and M. Jordan. Modeling annotated data. In Proc. International ACM SIGIR Conference on Research and Development in Information Retrieval, pages 127-134. ACM Press, 2003.

[4] D. Blei, A. Ng, and M. Jordan. Latent Dirichlet allocation. J. Machine Learning Research, 3:993-1022, Jan 2003.

[5] Z. Chen, L. Wenyin, F. Zhang and M. Li. Web mining for web image retrieval. J. of the American Society for Information Science and Technology, 51:10, pages 831-839, 2001.

[6] R. Fergus, L. Fei-Fei, P. Perona, and A. Zisserman. Learning Object Categories from Google's Image Search. In Proc. ICCV 2005.

[7] T. Griffiths and M. Steyvers. Finding Scientific Topics. In Proc. of the National Academy of Sciences, 101 (suppl. 1), pages 5228-5235, 2004.

[8] V. Jain, E. Learned-Miller, A. McCallum. People-LDA: Anchoring Topics to People using Face Recognition. In Proc. ICCV, 2007.

[9] J. Li, G. Wang, and L. Fei-Fei. OPTIMOL: automatic Object Picture collecTion via Incremental MOdel Learning. In Proc. CVPR, 2007.

[10] K. Mikolajczyk and C. Schmid. Scale and affine invariant interest point detectors. In Proc. IJCV, 2004.

[11] M. Porter, An algorithm for suffix stripping, Program, 14(3) pp 130-137, 1980.

[12] F. Schroff, A. Criminisi and A. Zisserman. Harvesting image databases from the web. In Proc. ICCV, 2007.

[13] M. Steyvers and T. Griffiths. Matlab Topic Modeling Toolbox.

   `http://psiexp.ss.uci.edu/research/programs_data/toolbox.htm`

[14] D. Yarowsky. Unsupervised word sense disambiguation rivaling supervised methods. ACL, 1995.

